# Learnability and the Doubling Dimension

**Yi Li**
Genome Institute of Singapore
liy3@gis.a-star.edu.sg

**Philip M. Long**
Google
plong@google.com

## Abstract

Given a set $F$ of classifiers and a probability distribution over their domain, one can define a metric by taking the distance between a pair of classifiers to be the probability that they classify a random item differently. We prove bounds on the sample complexity of PAC learning in terms of the doubling dimension of this metric. These bounds imply known bounds on the sample complexity of learning halfspaces with respect to the uniform distribution that are optimal up to a constant factor. We prove a bound that holds for any algorithm that outputs a classifier with zero error whenever this is possible; this bound is in terms of the maximum of the doubling dimension and the VC-dimension of $F$, and strengthens the best known bound in terms of the VC-dimension alone. We show that there is no bound on the doubling dimension in terms of the VC-dimension of $F$ (in contrast with the metric dimension).

## 1 Introduction

A set $F$ of classifiers and a probability distribution $D$ over their domain induce a metric in which the distance between classifiers is the probability that they disagree on how to classify a random object. (Let us call this metric $\rho_D$.) Properties of metrics like this have long been used for analyzing the generalization ability of learning algorithms [11, 32]. This paper is about bounds on the number of examples required for PAC learning in terms of the doubling dimension [4] of this metric space.

The doubling dimension of a metric space is the least $d$ such that any ball can be covered by $2^d$ balls of half its radius. The doubling dimension has been frequently used lately in the analysis of algorithms [13, 20, 21, 17, 29, 14, 7, 22, 28, 6].

In the PAC-learning model, an algorithm is given examples $(x_1, f(x_1)), ..., (x_m, f(x_m))$ of the behavior of an arbitrary member $f$ of a known class $F$. The items $x_1, ..., x_m$ are chosen independently at random according to $D$. The algorithm must, with probability at least $1 - \delta$ (w.r.t. to the random choice of $x_1, ..., x_m$), output a classifier whose distance from $f$ is at most $\epsilon$.

We show that if $(F, \rho_D)$ has doubling dimension $d$, then $F$ can be PAC-learned with respect to $D$ using

$$O\left(\frac{d + \log\frac{1}{\delta}}{\epsilon}\right) \tag{1}$$

examples. If in addition the VC-dimension of $F$ is $d$, we show that any algorithm that outputs a classifier with zero training error whenever this is possible PAC-learns $F$ w.r.t. $D$ using

$$O\left(\frac{d\sqrt{\log\frac{1}{\epsilon}} + \log\frac{1}{\delta}}{\epsilon}\right) \tag{2}$$

examples.

We show that if $F$ consists of halfspaces through the origin, and $D$ is the uniform distribution over the unit ball in $\mathbf{R}^n$, then the doubling dimension of $(F, \rho_D)$ is $O(n)$. Thus (1) generalizes the known bound of $O\left(\frac{n + \log \frac{1}{\delta}}{\epsilon}\right)$ for learning halfspaces with respect to the uniform distribution [25], matching a known lower bound for this problem [23] up to a constant factor. Both upper bounds improve on the $O\left(\frac{n \log \frac{1}{\epsilon} + \log \frac{1}{\delta}}{\epsilon}\right)$ bound that follows from the traditional analysis; (2) is the first such improvement for a polynomial-time algorithm.

Some previous analyses of the sample complexity of learning have made use of the fact that the "metric dimension" [18] is at most the VC-dimension [11, 15]. Since using the doubling dimension can sometimes lead to a better bound, a natural question is whether there is also a bound on the doubling dimension in terms of the VC-dimension. We show that this is not the case: it is possible to pack $(1/\alpha)^{(1/2 - o(1))d}$ classifiers in a set $F$ of VC-dimension $d$ so that the distance between every pair is in the interval $[\alpha, 2\alpha]$. Our analysis was inspired by some previous work in computational geometry [19], but is simpler.

Combining our upper bound analysis with established techniques (see [33, 3, 8, 31, 30]), one can perform similar analyses for the more general case in which no classifier in $F$ has zero error. We have begun with the PAC model because it is a clean setting in which to illustrate the power of the doubling dimension for analyzing learning algorithms. The doubling dimension appears most useful when the best achievable error rate (the Bayes error) is of the same order as the inverse of the number of training examples (or smaller).

Bounding the doubling dimension is useful for analyzing the sample complexity of learning because it limits the richness of a subclass of $F$ near the classifier to be learned. For other analyses that exploit bounds on such local richness, please see [31, 30, 5, 25, 26, 34]. It could be that stronger results could be obtained by marrying the techniques of this paper with those. In any case, it appears that the doubling dimension is an intuitive yet powerful way to bound the local complexity of a collection of classifiers.

## 2  Preliminaries

### 2.1  Learning

For some domain $X$, an *example* consists of a member of $X$, and its classification in $\{0, 1\}$. A *classifier* is a mapping from $X$ to $\{0, 1\}$. A *training set* is a finite collection of examples. A *learning algorithm* takes as input a training set, and outputs a classifier.

Suppose $D$ is a probability distribution over $X$. Then define

$$\rho_D(f, g) = \mathbf{Pr}_{x \sim D}(f(x) \neq g(x)).$$

A learning algorithm $A$ PAC learns $F$ w.r.t. $D$ with accuracy $\epsilon$ and confidence $\delta$ from $m$ examples if, for any $f \in F$, if

- domain elements $x_1, ..., x_m$ are drawn independently at random according to $D$, and
- $(x_1, f(x_1)), ..., (x_m, f(x_m))$ is passed to $A$, which outputs $h$,

then

$$\mathbf{Pr}(\rho_D(f, h) > \epsilon) \leq \delta.$$

If $F$ is a set of classifiers, a learning algorithm is a *consistent hypothesis finder for $F$* if it outputs an element of $F$ that correctly classifies all of the training data whenever it is possible to do so.

### 2.2  Metrics

Suppose $\Phi = (Z, \rho)$ is a metric space.

An $\alpha$-cover for $\Phi$ is a set $T \subseteq Z$ such that every element of $Z$ has a counterpart in $T$ that is at a distance at most $\alpha$ (with respect to $\rho$).

An $\alpha$-packing for $\Phi$ is a set $T \subseteq Z$ such that every pair of elements of $T$ are at a distance greater than $\alpha$ (again, with respect to $\rho$).

The $\alpha$-ball centered at $z \in Z$ consists of all $t \in Z$ for which $\rho(z, t) \leq \alpha$.

Denote the size of the smallest $\alpha$-cover by $\mathcal{N}(\alpha; \Phi)$. Denote the size of the largest $\alpha$-packing by $\mathcal{M}(\alpha; \Phi)$.

**Lemma 1 ([18])** *For any metric space $\Phi = (Z, \rho)$, and any $\alpha > 0$,*

$$\mathcal{M}(2\alpha; \Phi) \leq \mathcal{N}(\alpha; \Phi) \leq \mathcal{M}(\alpha; \Phi).$$

The *doubling dimension* of $\Phi$ is the least $d$ such that, for all radii $\alpha > 0$, any $\alpha$-ball in $\Phi$ can be covered by at most $2^d$ $\alpha/2$-balls. That is, for any $\alpha > 0$ and any $z \in Z$, there is a $C \subseteq Z$ such that

- $|C| \leq 2^d$, and
- $\{t \in Z : \rho(z, t) \leq \alpha\} \subseteq \cup_{c \in C} \{t \in Z : \rho(c, t) \leq \alpha/2\}$.

### 2.3 Probability

For a function $\psi$ and a probability distribution $D$, let $\mathbf{E}_{x \sim D}(\psi(x))$ be the expectation of $\psi$ w.r.t. $D$. We will shorten this to $\mathbf{E}_D(\psi)$, and if $\mathbf{u} = (u_1, ..., u_m) \in X^m$, then $\mathbf{E}_{\mathbf{u}}(\psi)$ will be $\frac{1}{m} \sum_{i=1}^m \psi(u_i)$. We will use $\mathbf{Pr}_{x \sim D}$, $\mathbf{Pr}_D$, and $\mathbf{Pr}_{\mathbf{u}}$ similarly.

## 3 The strongest upper bound

**Theorem 2** *Suppose $d$ is the doubling dimension of $(F, \rho_D)$. There is an algorithm $A$ that PAC-learns $F$ from $O\left(\frac{d + \log(1/\delta)}{\epsilon}\right)$ examples.*

The key lemma limits the extent to which points that are separated from one another can crowd around some point in a metric space with limited doubling dimension.

**Lemma 3 (see [13])** *Suppose $\Phi = (Z, \rho)$ is a metric space with doubling dimension $d$ and $z \in Z$.*

*For $\beta > 0$, let $B(z, \beta)$ consist of the elements of $u \in Z$ such that $\rho(u, z) \leq \beta$ (that is, the $\beta$-ball centered at $z$). Then*

$$\mathcal{M}(\alpha, B(z, \beta)) \leq \left(\frac{8\beta}{\alpha}\right)^d.$$

*(In other words, any $\alpha$-packing must have at most $(8\beta/\alpha)^d$ elements within distance $\beta$ of $z$.)*

**Proof**: Since $\Phi$ has doubling dimension $d$, the set $B(z, \beta)$ can be covered by $2^d$ balls of radius $\beta/2$. Each of these can be covered by $2^d$ balls of radius $\beta/4$, and so on. Thus, $B(z, \beta)$ can be covered by $2^{d \lceil \log_2 \beta/\alpha \rceil} \leq (4\beta/\alpha)^d$ balls of radius $\alpha/2$. Applying Lemma 1 completes the proof. ∎

Now we are ready to prove Theorem 2. The proof is an application of the peeling technique [1] (see [30]).

**Proof of Theorem 2**: Construct an $\epsilon/4$ packing $G$ greedily, by repeatedly adding an element of $F$ to $G$ for as long as this is possible. This packing is also an $\epsilon/4$-cover, since otherwise we could add another member to $G$.

Consider the algorithm that outputs the element of $G$ with minimum error on the training set. Whatever the target, some element of $G$ has error at most $\epsilon/4$. Applying Chernoff bounds, $O\left(\frac{\log(1/\delta)}{\epsilon}\right)$ examples are sufficient that, with probability at least $1 - \delta/2$, this classifier is incorrect on at most a fraction $\epsilon/2$ of the training data. Thus, the training error of the hypothesis output by $A$ is at most $\epsilon/2$ with probability at least $1 - \delta/2$.

Choose an arbitrary function $f$, and let $S$ be the random training set resulting from drawing $m$ examples according to $D$, and classifying them using $f$. Define $\rho_S(g, h)$ to be the fraction of examples

in $S$ on which $g$ and $h$ disagree. We have

$$\mathbf{Pr}(\exists g \in G, \ \rho_D(g,f) > \epsilon \text{ and } \rho_S(g,f) \leq \epsilon/2)$$

$$\leq \sum_{k=0}^{\log(1/\epsilon)} \mathbf{Pr}(\exists g \in G, \ 2^k \epsilon < \rho_D(g,f) \leq 2^{k+1}\epsilon \text{ and } \rho_S(g,f) \leq \epsilon/2)$$

$$\leq \sum_{k=0}^{\log(1/\epsilon)} 2^{(k+5)d} e^{-2^k \epsilon m/8}$$

by Lemma 3 and the standard Chernoff bound.

Each of the following steps is a straightforward manipulation:

$$\sum_{k=0}^{\log(1/\epsilon)} 2^{(k+5)d} e^{-2^k \epsilon m/8} = 32^d \sum_{k=0}^{\log(1/\epsilon)} 2^{kd} e^{-2^k \epsilon m/8} \leq 32^d \sum_{k=0}^{\log(1/\epsilon)} 2^{2^k d} e^{-2^k \epsilon m/8}$$

$$\leq 32^d \sum_{k=0}^{\infty} 2^{2^k d} e^{-2^k \epsilon m/8} \leq \frac{64^d e^{-\epsilon m/8}}{1 - 2^d e^{-\epsilon m/8}}.$$

Since $m = O((d + \log(1/\delta))/\epsilon)$ is sufficient for $64^d e^{-\epsilon m/8} \leq \delta/2$ and $2^d e^{-\epsilon m/8} \leq 1/2$, this completes the proof. ∎

## 4  A bound for consistent hypothesis finders

In this section we analyze algorithms that work by finding hypotheses with zero training error. This is one way to achieve computational efficiency, as is the case when $F$ consists of halfspaces. This analysis will use the notion of VC-dimension.

**Definition 4** *The VC-dimension of a set $F$ of $\{0,1\}$-valued functions with a common domain is the size of the largest set $x_1, ..., x_d$ of domain elements such that*

$$\{(f(x_1), ..., f(x_d)) : f \in F\} = \{0,1\}^d.$$

The following lemma generalizes the Chernoff bound to hold uniformly over a class of random variables; it concentrates on a simplified consequence of the Chernoff bound that is useful when bounding the probability that an empirical estimate is *much* larger than the true expectation.

**Lemma 5 (see [12, 24])** *Suppose $F$ is a set of $\{0,1\}$-valued functions with a common domain $X$. Let $d$ be the VC-dimension of $F$. Let $D$ be a probability distribution over $X$. Choose $\alpha > 0$ and $K \geq 1$. Then if*

$$m \geq \frac{c(d \log \frac{1}{\alpha} + \log \frac{1}{\delta})}{\alpha K \log(1 + K)},$$

*where $c$ is an absolute constant, then*

$$\mathbf{Pr}_{\mathbf{u} \sim D^m}(\exists f, g \in F, \ \mathbf{Pr}_D(f \neq g) \leq \alpha \text{ but } \mathbf{Pr}_{\mathbf{u}}(f \neq g) > (1 + K)\alpha) \leq \delta.$$

Now we are ready for the main analysis of this section.

**Theorem 6** *Suppose the doubling dimension of $(F, \rho_D)$ and the VC-dimension of $F$ are both at most $d$. Any consistent hypothesis finder for $F$ PAC learns $F$ from $O\left(\frac{1}{\epsilon}\left(d\sqrt{\log \frac{1}{\epsilon}} + \log \frac{1}{\delta}\right)\right)$ examples.*

**Proof**: Assume without loss of generality that $\epsilon \leq 1/100$. Let $\alpha = \epsilon \exp(-\sqrt{\ln \frac{1}{\epsilon}})$; since $\epsilon \leq 1/100$, we have $\alpha \leq \epsilon/8$.

Choose a target function $f$. For each $h \in F$, define $\ell_h : X \to \{0,1\}$ by $\ell_h(x) = 1 \Leftrightarrow h(x) \neq f(x)$. Let $\ell_F = \{\ell_h : h \in F\}$. Since $\ell_g(x) \neq \ell_h(x)$ exactly when $g(x) \neq h(x)$, the doubling dimension

of $\ell_F$ is the same as the doubling dimension of $F$; the VC-dimension of $\ell_F$ is also known to be the same as the VC-dimension of $F$ (see [32]).

Construct an $\alpha$ packing $G$ greedily, by repeatedly adding an element of $\ell_F$ to $G$ for as long as this is possible. This packing is also an $\alpha$-cover.

For each $g \in \ell_F$, let $\phi(g)$ be its nearest neighbor in $G$. Since $\alpha \le \epsilon/8$, by the triangle inequality,

$$\mathbf{E}_D(g) > \epsilon \text{ and } \mathbf{E_u}(g) = 0 \ \Rightarrow \ \mathbf{E}_D(\phi(g)) > 7\epsilon/8 \text{ and } \mathbf{E_u}(g) = 0. \tag{3}$$

The triangle inequality also yields

$$\mathbf{E_u}(g) = 0 \ \Rightarrow \ (\mathbf{E_u}(\phi(g)) \le \epsilon/4 \text{ or } \mathbf{Pr_u}(\phi(g) \ne g) > \epsilon/4).$$

Combining this with (3), we have we have

$$\begin{aligned}
&\mathbf{Pr}(\exists g \in \ell_F, \mathbf{E}_D(g) > \epsilon \text{ but } \mathbf{E_u}(g) = 0) \\
&\le \mathbf{Pr}(\exists g \in \ell_F, \mathbf{E}_D(\phi(g)) > 7\epsilon/8 \text{ but } \mathbf{E_u}(\phi(g)) \le \epsilon/4) \\
&\qquad + \mathbf{Pr}(\exists g \in \ell_F, \mathbf{Pr_u}(\phi(g) \ne g) > \epsilon/4).
\end{aligned} \tag{4}$$

We have

$$\begin{aligned}
&\mathbf{Pr}(\exists g \in \ell_F, \mathbf{E}_D(\phi(g)) > 7\epsilon/8 \text{ but } \mathbf{E_u}(\phi(g)) \le \epsilon/4) \\
&\le \mathbf{Pr}(\exists g \in G, \mathbf{E}_D(g) > 7\epsilon/8 \text{ but } \mathbf{E_u}(g) \le \epsilon/4) \\
&= \mathbf{Pr}(\exists g \in G, \rho_D(f,g) > 7\epsilon/8 \text{ but } \mathbf{Pr_u}(f \ne g) \le \epsilon/4) \\
&\le \sum_{k=0}^{\log(8/(7\epsilon))} \mathbf{Pr}(\exists g \in G, \ 2^k(7\epsilon/8) < \rho_D(g,f) \le 2^{k+1}(7\epsilon/8) \text{ and } \mathbf{Pr_u}(f \ne g) \le \epsilon/4) \\
&\le \sum_{k=0}^{\log(8/(7\epsilon))} \left( \frac{8\epsilon 2^{k+1}}{\alpha} \right)^d e^{-c 2^k \epsilon m},
\end{aligned}$$

where $c > 0$ is an absolute constant, by Lemma 3 and the standard Chernoff bound.

Computing a geometric sum exactly as in the proof of Theorem 2, we have that $m = O(d/\epsilon)$ suffices for

$$\mathbf{Pr}(\exists g \in \ell_F, \mathbf{E}_D(\phi(g)) > 7\epsilon/8 \text{ but } \mathbf{E_u}(\phi(g)) \le \epsilon/4) \le \left( \frac{c_1 \epsilon}{\alpha} \right)^d e^{-c_2 \epsilon m},$$

for absolute constants $c_1, c_2 > 0$.

By plugging in the value of $\alpha$ and solving, we can see that

$$m = O\left( \frac{1}{\epsilon}\left( d\sqrt{\log \frac{1}{\epsilon}} + \log \frac{1}{\delta} \right) \right)$$

suffices for

$$\mathbf{Pr}(\exists g \in \ell_F, \mathbf{Pr}_D(\phi(g)) > 7\epsilon/8 \text{ but } \mathbf{Pr_u}(\phi(g)) \le \epsilon/4) \le \delta/2. \tag{5}$$

Since $\mathbf{Pr}_D(\phi(g) \ne g) \le \alpha \le \epsilon/8$ for all $g \in \ell_F$, applying Lemma 5 with $K = \epsilon/(4\alpha) - 1$, we get that there is an absolute constant $c > 0$ such that

$$m \ge \frac{c\left( d\log\frac{1}{\alpha} + \log\frac{1}{\delta} \right)}{(\epsilon/4 - \alpha)\log(\frac{\epsilon}{4\alpha})} \tag{6}$$

also suffices for

$$\mathbf{Pr}(\exists g \in \ell_F, \mathbf{Pr_u}(\phi(g) \ne g) > \epsilon/4) \le \delta/2.$$

Substituting the value $\alpha$ into (6), it is sufficient that

$$m \ge \frac{c\left( d(\log\frac{1}{\epsilon} + \sqrt{\log\frac{1}{\epsilon}}) + \log\frac{1}{\delta} \right)}{(\epsilon/8)\sqrt{\log\frac{1}{\epsilon} - \log 4}}.$$

Putting this together with (5) and (4) completes the proof. ∎

# 5 Halfspaces and the uniform distribution

**Proposition 7** *If $U_n$ is the uniform distribution over the unit ball in $\mathbf{R}^n$, and $H_n$ is the set of halfspaces that go through the origin, then the doubling dimension of $(H_n, \rho_{U_n})$ is $O(n)$.*

**Proof**: Choose $h \in H_n$ and $\alpha > 0$. We will show that the ball of radius $\alpha$ centered at $h$ can be covered by $O(n)$ balls of radius $\alpha/2$.

Suppose $U_{H_n}$ is the probability distribution over $H_n$ obtained by choosing a normal vector $\mathbf{w}$ uniformly from the unit ball, and outputting $\{\mathbf{x} : \mathbf{w} \cdot \mathbf{x} \geq 0\}$. The argument will be a "volume argument" using $U_{H_n}$.

It is known (see Lemma 4 of [25]) that

$$\mathbf{Pr}_{g \sim U_{H_n}} (\rho_{U_n}(g,h) \leq \alpha/4) \geq (c_1 \alpha)^{n-1}$$

where $c_1 > 0$ is an absolute constant independent of $\alpha$ and $n$. Furthermore,

$$\mathbf{Pr}_{g \sim U_{H_n}} (\rho_{U_n}(g,h) \leq 5\alpha/4) \leq (c_2 \alpha)^{n-1}$$

where $c_2 > 0$ is another absolute constant.

Suppose we choose arbitrarily choose $g_1, g_2, \ldots \in H_n$ that are at a distance at most $\alpha$ from $h$, but $\alpha/2$ far from one another. By the triangle inequality, $\alpha/4$-balls centered at $g_1, g_2, \ldots$ are disjoint. Thus, the probability that an random element of $H_n$ is in a ball of radius $\alpha/4$ centered at one of $g_1, \ldots, g_N$ is at least $N(c_1 \alpha)^{n-1}$. On the other hand, since each $g_1, \ldots, g_N$ has distance at most $\alpha$ from $h$, any element of an $\alpha/4$ ball centered at one of them is at most $\alpha + \alpha/4$ far from $h$. Thus, the union of the $\alpha/4$ balls centered at $g_1, \ldots, g_N$ is contained in the $5\alpha/4$ ball centered at $h$. Thus $N(c_1 \alpha)^{n-1} \leq (c_2 \alpha)^{n-1}$, which implies $N \leq (c_2/c_1)^{n-1} = 2^{O(n)}$, completing the proof. ∎

# 6 Separation

**Theorem 8** *For all $\alpha \in [0, 1/2]$ and positive integers $d$ there is a set $F$ of classifiers and a probability distribution $D$ over their common domain with the following properties:*

- *the VC-dimension of $F$ is $d$*

- *$|F| \geq \lfloor \frac{1}{2} \left( \frac{1}{2e\alpha} \right)^{d/2} \rfloor$*

- *for each $f, g \in F$, $\alpha \leq \rho_D(f, g) \leq 2\alpha$.*

This proof uses the probabilistic method. We begin with the following lemma.

**Lemma 9** *Choose positive integers $s$ and $d$. Suppose $A$ is chosen uniformly at random from among the subsets of $\{1, \ldots, s\}$ of size $d$. Then, for any $B > 1$,*

$$\mathbf{Pr}(|A \cap \{1, \ldots, d\}| \geq (1+B)E(|A \cap \{1, \ldots, d\}|)) \leq \left( \frac{e}{1+B} \right)^{(1+B)E(|A \cap \{1,\ldots,d\}|)} .$$

**Proof**: in Appendix A. ∎

Now we're ready for the proof of Theorem 8, which uses the deletion technique (see [2]).

**Proof** (of Theorem 8): Set the domain $X$ to be $\{1, \ldots, s\}$, where $s = \lceil d/\alpha \rceil$. Let $N = \lfloor \left( \frac{s}{2ed} \right)^{d/2} \rfloor$. Suppose $f_1, \ldots, f_N$ are chosen independently, uniformly at random from among the classifiers that evaluate to 1 on exactly $d$ elements of $X$. For any distinct $i, j$, suppose $f_i^{-1}(1)$ is fixed, and we think of the members of $f_j^{-1}(1)$ as being chosen one at a time. The probability that any of the elements of $f_j^{-1}(1)$ is also in $f_i^{-1}(1)$ is $d/s$. Applying the linearity of expectation, and averaging over the different possibilities for $f_i^{-1}(1)$, we get

$$\mathbf{E}(|f_i^{-1}(1) \cap f_j^{-1}(1)|) = \frac{d^2}{s}.$$

Applying Lemma 9,

$$
\begin{aligned}
\mathbf{Pr}(|f_i^{-1}(1) \cap f_j^{-1}(1)| \geq d/2) &\leq \left(\frac{2ed}{s}\right)^{\left(\frac{s}{2d}\right)\left(\frac{d^2}{s}\right)} \\
&= \left(\frac{2ed}{s}\right)^{d/2}.
\end{aligned}
$$

Thus, the expected number of pairs $i, j$ such that $\mathbf{Pr}(|f_i^{-1}(1) \cap f_j^{-1}(1)| \geq d/2)$ is at most $(N^2/2)\left(\frac{2ed}{s}\right)^{d/2}$. This implies that there exist $f_1, ..., f_N$ such that

$$
|\{\{i, j\} : |f_i^{-1}(1) \cap f_j^{-1}(1)| \geq d/2\}| \leq (N^2/2)\left(\frac{2ed}{s}\right)^{d/2}.
$$

If we delete one element from each such pair, and form $G$ from what remains, then each pair $g, h$ of elements in $G$ satisfies

$$
|g^{-1}(1) \cap h^{-1}(1)| < d/2. \tag{7}
$$

If $D$ is the uniform distribution over $\{1, ..., s\}$, then (7) implies $\rho_D(g, h) > \alpha$. The number of elements of $G$ is at least $N - (N^2/2)\left(\frac{2ed}{s}\right)^{d/2} \geq N/2$.

Since each $g \in G$ has $g^{-1}(1) = d$, no function in $G$ evaluates to 1 on each element of any set of $d + 1$ elements of $X$. Thus, the VC-dimension of $G$ is at most $d$. ∎

Theorem 8 implies that there is no bound on the doubling dimension of $(G, \rho_D)$ in terms of the VC-dimension of $G$. For any constraint on the VC-dimension, a set $G$ satisfying the constraint can have arbitrarily large doubling dimension by setting the value of $\alpha$ in Theorem 8 arbitrarily small.

## Acknowledgement

We thank Gábor Lugosi and Tong Zhang for their help.

## References

[1] K. Alexander. Rates of growth for weighted empirical processes. In *Proc. of Berkeley Conference in Honor of Jerzy Neyman and Jack Kiefer*, volume 2, pages 475–493, 1985.

[2] N. Alon, J. H. Spencer, and P. Erdös. *The Probabilistic Method.* Wiley, 1992.

[3] M. Anthony and P. L. Bartlett. *Neural Network Learning: Theoretical Foundations.* Cambridge University Press, 1999.

[4] P. Assouad. Plongements lipschitziens dans. $R$. *Bull. Soc. Math. France*, 111(4):429–448, 1983.

[5] P. L. Bartlett, O. Bousquet, and S. Mendelson. Local Rademacher complexities. *Annals of Statistics*, 33(4):1497–1537, 2005.

[6] A. Beygelzimer, S. Kakade, and J. Langford. Cover trees for nearest neighbor. *ICML*, 2006.

[7] H. T. H. Chan, A. Gupta, B. M. Maggs, and S. Zhou. On hierarchical routing in doubling metrics. *SODA*, 2005.

[8] L. Devroye, L. Györfi, and G. Lugosi. *A Probabilistic Theory of Pattern Recognition.* Springer, 1996.

[9] D. Dubhashi and D. Ranjan. Balls and bins: A study in negative dependence. *Random Structures & Algorithms*, 13(2):99–124, Sept 1998.

[10] Devdatt Dubhashi, Volker Priebe, and Desh Ranjan. Negative dependence through the FKG inequality. Technical Report RS-96-27, BRICS, 1996.

[11] R. M. Dudley. Central limit theorems for empirical measures. *Annals of Probability*, 6(6):899–929, 1978.

[12] R. M. Dudley. A course on empirical processes. *Lecture notes in mathematics*, 1097:2–142, 1984.

[13] A. Gupta, R. Krauthgamer, and J. R. Lee. Bounded geometries, fractals, and low-distortion embeddings. *FOCS*, 2003.

[14] S. Har-Peled and M. Mendel. Fast construction of nets in low dimensional metrics, and their applications. *SICOMP*, 35(5):1148–1184, 2006.

[15] D. Haussler. Sphere packing numbers for subsets of the Boolean $n$-cube with bounded Vapnik-Chervonenkis dimension. *Journal of Combinatorial Theory, Series A*, 69(2):217–232, 1995.

[16] K. Joag-Dev and F. Proschan. Negative association of random variables, with applications. *The Annals of Statistics*, 11(1):286–295, 1983.

[17] J. Kleinberg, A. Slivkins, and T. Wexler. Triangulation and embedding using small sets of beacons. *FOCS*, 2004.

[18] A. N. Kolmogorov and V. M. Tihomirov. $\epsilon$-entropy and $\epsilon$-capacity of sets in functional spaces. *American Mathematical Society Translations (Ser. 2)*, 17:277–364, 1961.

[19] J. Komlós, J. Pach, and G. Woeginger. Almost tight bounds on epsilon-nets. *Discrete and Computational Geometry*, 7:163–173, 1992.

[20] R. Krauthgamer and J. R. Lee. The black-box complexity of nearest neighbor search. *ICALP*, 2004.

[21] R. Krauthgamer and J. R. Lee. Navigating nets: simple algorithms for proximity search. *SODA*, 2004.

[22] F. Kuhn, T. Moscibroda, and R. Wattenhofer. On the locality of bounded growth. *PODC*, 2005.

[23] P. M. Long. On the sample complexity of PAC learning halfspaces against the uniform distribution. *IEEE Transactions on Neural Networks*, 6(6):1556–1559, 1995.

[24] P. M. Long. Using the pseudo-dimension to analyze approximation algorithms for integer programming. *Proceedings of the Seventh International Workshop on Algorithms and Data Structures*, 2001.

[25] P. M. Long. An upper bound on the sample complexity of PAC learning halfspaces with respect to the uniform distribution. *Information Processing Letters*, 87(5):229–234, 2003.

[26] S. Mendelson. Estimating the performance of kernel classes. *Journal of Machine Learning Research*, 4:759–771, 2003.

[27] R. Motwani and P. Raghavan. *Randomized Algorithms*. Cambridge University Press, 1995.

[28] A. Slivkins. Distance estimation and object location via rings of neighbors. *PODC*, 2005.

[29] K. Talwar. Bypassing the embedding: Approximation schemes and compact representations for low dimensional metrics. *STOC*, 2004.

[30] S. van de Geer. *Empirical processes in M-estimation*. Cambridge Series in Statistical and Probabilistic Methods, 2000.

[31] A. van der Vaart and J. A. Wellner. *Weak Convergence and Empirical Processes With Applications to Statistics*. Springer, 1996.

[32] V. N. Vapnik. *Estimation of Dependencies based on Empirical Data*. Springer Verlag, 1982.

[33] V. N. Vapnik. *Statistical Learning Theory*. New York, 1998.

[34] T. Zhang. Information theoretical upper and lower bounds for statistical estimation. *IEEE Transactions on Information Theory*, 2006. to appear.

# A    Proof of Lemma 9

**Definition 10 ([16])** *A collection $X_1, ..., X_n$ of random variables are* negatively associated *if for every disjoint pair $I, J \subseteq \{1, ..., n\}$ of index sets, and for every pair $f : \mathbf{R}^{|I|} \to \mathbf{R}$ and $g : \mathbf{R}^{|J|} \to \mathbf{R}$ of non-decreasing functions, we have*

$$\mathbf{E}(f(X_i, i \in I)g(X_j, j \in J)) \leq \mathbf{E}(f(X_i, i \in I))\mathbf{E}(g(X_j, j \in J)).$$

**Lemma 11 ([10])** *If $A$ is chosen uniformly at random from among the subsets of $\{1, ..., s\}$ with exactly $d$ elements, and $X_i = 1$ if $i \in A$ and $0$ otherwise, then $X_1, ..., X_s$ are negatively associated.*

**Lemma 12 ([9])** *Collections $X_1, ..., X_n$ of negatively associated random variables satisfy Chernoff bounds: for any $\lambda > 0$, $\mathbf{E}(\exp(\lambda \sum_{i=1}^{n} X_i)) \leq \prod_{i=1}^{n} \mathbf{E}(\exp(\lambda X_i))$.*

**Proof of Lemma 9**: Let $X_i \in \{0, 1\}$ indicate whether $i \in A$. By Lemma 11, $X_1, ..., X_d$ are negatively associated. We have $|\{A \cap \{1, ..., d\}\}| = \sum_{i=1}^{d} X_i$. Combining Lemma 12 with a standard Chernoff-Hoeffding bound (see Theorem 4.1 of [27]) completes the proof. ∎